# Constructing Hidden Units using Examples and Queries

**Eric B. Baum**     **Kevin J. Lang**
NEC Research Institute
4 Independence Way
Princeton, NJ 08540

## ABSTRACT

While the network loading problem for 2-layer threshold nets is NP-hard when learning from examples alone (as with backpropagation), (Baum, 91) has now proved that a learner can employ queries to evade the hidden unit credit assignment problem and PAC-load nets with up to four hidden units in polynomial time. Empirical tests show that the method can also learn far more complicated functions such as randomly generated networks with 200 hidden units. The algorithm easily approximates Wieland's 2-spirals function using a single layer of 50 hidden units, and requires only 30 minutes of CPU time to learn 200-bit parity to 99.7% accuracy.

## 1   Introduction

Recent theoretical results (Baum & Haussler, 89) promise good generalization from multi-layer feedforward nets that are consistent with sufficiently large training sets. Unfortunately, the problem of finding such a net has been proved intractable due to the hidden unit credit assignment problem — even for nets containing only 2 hidden units (Blum & Rivest, 88). While back-propagation works well enough on simple problems, its luck runs out on tasks requiring more than a handful of hidden units. Consider, for example, Alexis Wielands "2-spirals" mapping from $\Re^2$ to $\{0, 1\}$. There are many sets of weights that would cause a 2-50-1 network to be consistent with the training set of figure 3a, but backpropagation seems unable to find any of them starting from random initial weights. Instead, the procedure drives the net into a suboptimal configuration like the one pictured in figure 2b.

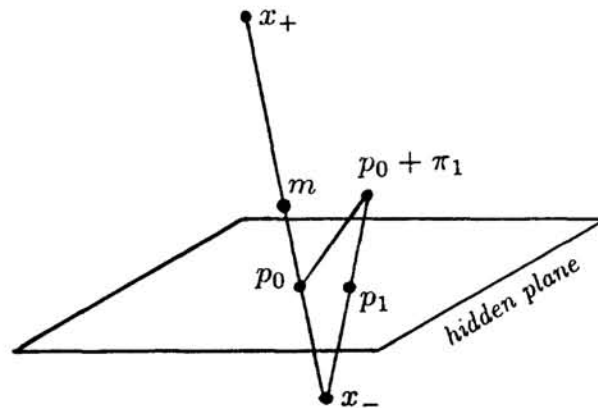

**Figure 1:** The geometry of query learning.

In 1984, Valiant proposed a *query learning* model in which the learner can ask an oracle for the output values associated with arbitrary points in the input space. In the next section we shall see how this additional source of information can be exploited to locate and pin down a network's hidden units one at a time, thus avoiding the combinatorial explosion of possible hidden unit configurations which can arise when one attempts to learn from examples alone.

## 2   How to find a hidden unit using queries

For now, assume that our task is to build a 2-layer network of binary threshold units which computes the same function as an existing "target" network. Our first step will be to draw a positive example $x_+$ and a negative example $x_-$ from our training set. Because the target net maps these points to different output values, its hidden layer representations for the points must also be different, so the hyperplane through input space corresponding to one of the net's hidden units must intersect the line segment bounded by the two points (see figure 1). We can reduce our uncertainty about the location of this intersection point by a factor of 2 by asking the oracle for the target net's output at $m$, the line segment's midpoint. If, for example, $m$ is mapped to the same output as $x_+$, then we know that the hidden plane must intersect the line segment between $x_-$ and $m$, and we can then further reduce our uncertainty by querying the midpoint of *this* segment. By performing $b$ of queries of this sort, we can determine to within b bits of accuracy the location of a point $p_0$ that lies on the hidden plane. Assuming that our input space has $n$ dimensions, after finding $n - 1$ more points on this hyperplane we can solve $n$ equations in $n$ unknowns to find the weights of the corresponding hidden unit.[1]

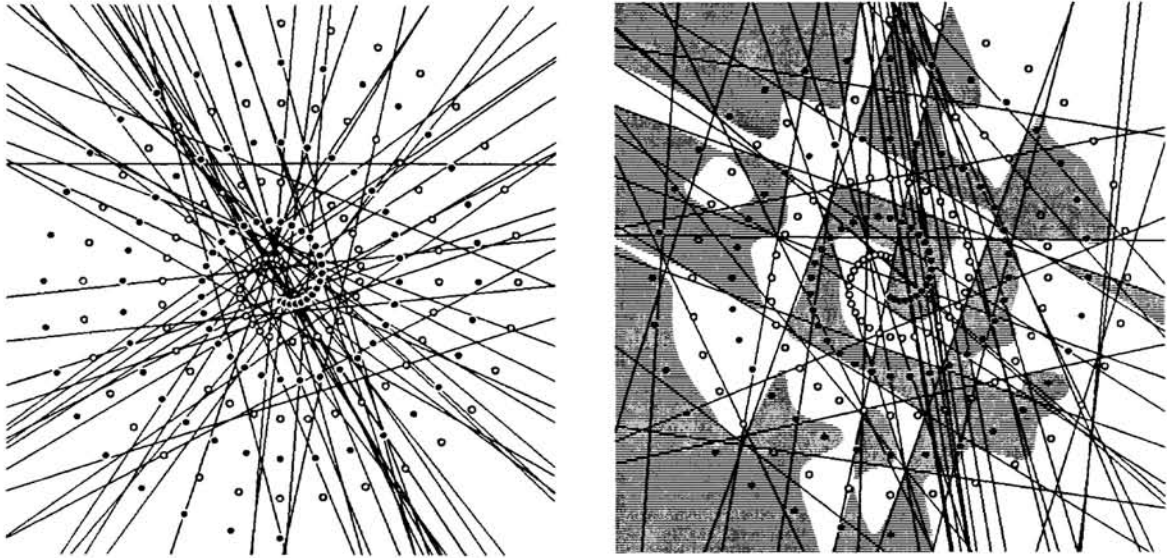

**Figure 2:** A backprop net before and after being trained on the 2-spirals task. In these plots over input space, the net's hidden units are shown by lines while its output is indicated by grey-level shading.

## 3    Can we find all of a network's hidden units?

Here is the crucial question: now that we have a procedure for finding *one* hidden unit whose hyperplane passes between a given pair of positive and negative examples,[2] can we discover *all* of the net's hidden units by invoking this procedure on a sequence of such example pairs? If the answer is yes, then we have got a viable learning method because the net's output weights can be efficiently computed via the linear programming problem that arises from forward-propagating the training set through the net's first layer of weights. (Baum, 91) proves that for target nets with four or fewer hidden units we can always find enough of them to compute the required function. This result is a direct counterpoint to the theorem of (Blum & Rivest, 88): by using queries, we can PAC learn in polynomial time small threshold nets that would be NP-hard to learn from examples alone.

However, it is possible for an adversary to construct a larger target net and an input distribution such that we may not find enough hidden units to compute the target function even by searching between every pair of examples in our training set. The problem is that more than one hidden plane can pass between a given pair of points, so we could repeatedly encounter some of the hidden units while never seeing others.

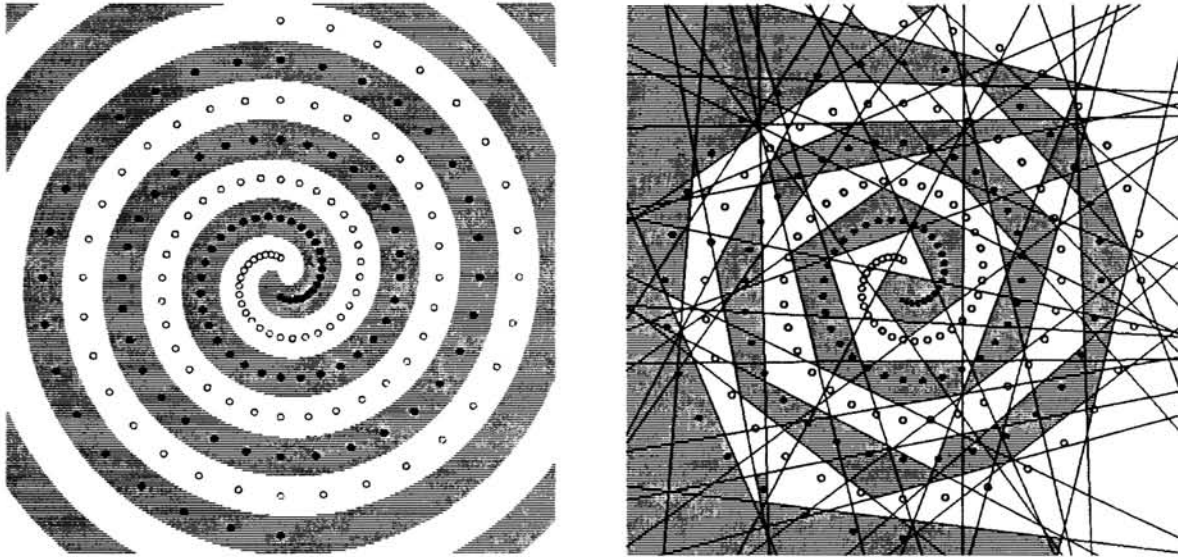

**Figure 3:** 2-spirals oracle, and net built by query learning.

Fortunately, the experiments described in the next section suggest that one can find most of a net's hidden units in the average case. In fact, we may not even need to find all of a network's hidden units in order to achieve good generalization. Suppose that one of a network's hidden units is hard to find due to the rarity of nearby training points. As long as our test set is drawn from the same distribution as the training set, examples that would be misclassified due to the absence of this plane will also be rare. Our experiment on learning 200-bit parity illustrates this point: only 1/4 of the possible hidden units were needed to achieve 99.7% generalization.

## 4   Learning random target nets

Although query learning might fail to discover hidden units in the worse case, the following empirical study suggests that the method has good behavior in the average case. In each of these learning experiments the target function was computed by a 2-layer threshold net whose $k$ hidden units were each chosen by passing a hyperplane through a set of $n$ points selected from the uniform distribution on the unit $n$-sphere. The output weights of each target net corresponded to a random hyperplane through the origin of the unit $k$-sphere. Our training examples were drawn from the uniform distribution on the corners of the unit $n$-cube and then classified according to the target net.

To establish a performance baseline, we attempted to learn several of these functions using backpropagation. For $(n = 20,\ k = 20)$ we succeeded in training a net to 97% accuracy in less than a day, but when we increased the size of the problem to $(n = 100,\ k = 50)$ or $(n = 200,\ k = 30)$, 150 hours of CPU time dumped our backprop nets into local minima that accounted for only 90% of the training data.

In contrast, query learning required only 1.5 hours to learn either of the latter two functions to 99% accuracy. The method continued to function well when we increased the problem size to ($n = 200$, $k = 200$). In each of five trials at this scale, a check of $10^4$ training pairs revealed 197 or more hidden planes. Because the networks were missing a couple of hidden units, their hidden-to-output mappings were not quite linearly separable. Nevertheless, by running the perceptron algorithm on $100 \times k$ random examples, in each trial we obtained approximate output weights whose generalization was 98% or better.

## 5    Learning 200-bit parity

Because the learning method described above needs to make real-valued queries in order to localize a hidden plane, it cannot be used to learn a function that is only defined on boolean inputs. Thus, we defined the parity of a real-valued vector to be the function computed by the 2-layer parity net of (Rumelhart, Hinton & Williams, 1986), which has input weights of 1, hidden unit thresholds of $\frac{1}{2}, \frac{3}{2}, ..., n - \frac{1}{2}$, and output weights alternating between 1 and $-1$. The $n$ parallel hidden planes of this net carve the input space into $n + 1$ diagonal slabs, each of which contains all of the binary patterns with a particular number of 1's.

After adopting this definition of parity (which agrees with the standard definition on boolean inputs), we applied the query learning algorithm to 200-dimensional input patterns. A search of 30,000 pairs of examples drawn randomly and uniformly from the corners of the unit cube revealed 46 of the 200 decision planes of the target function. Using approximate output weights computed by the perceptron algorithm, we found the nets generalization rate to be 99.7%. If it seems surprising that the net could perform so well while lacking so many hidden planes, consider the following. The target planes that we did find were the middle ones with thresholds near 100, and these are the relevant ones for classifying inputs that contain about the same number of 1's and 0's. Because vectors of uniform random bits are unlikely to contain many more 1's than 0's or *vice versa*, we had little chance of stumbling across hidden planes with high or low thresholds while learning, but we were also unlikely to need them for classifying any given test case.

## 6    Function approximation using queries

Suppose now that our goal in building a threshold net is to approximate an arbitrary function rather than to duplicate an existing threshold net. Earlier, we were worried about whether we could locate all of a target net's hidden units, but at least we knew how many of them there were, and we knew that we had made real progress when we found one of them. Now, the hidden units constructed by our algorithm are merely tangents to the true decision boundaries of the target fuction, and we do not know ahead of time how many such units will be required to construct a decent approximation to the function.

While one could keep adding hidden units to a net until the hidden layer representation of the training set becomes linearly separable, the fact that there are

| learning | additional | hidden units | | train | test errors | |
|---|---|---|---|---|---|---|
| algorithm | heuristics | min | max | errors | min | max |
| | none | 90 | 160 | 0 | 70 | 136 |
| queries | reject redundant units | 65 | 80 | 0 | 47 | 72 |
| | two-stage construction | 49 | 59 | 0 | 15 | 45 |
| conjugate gradient backprop | | 60 | | avg=9 | 80 | 125 |

**Table 1:** 2-spirals performance summary.

infinitely many of tangents to a given curve can result in the creation of an over-sized net which generalizes poorly. This problem can be addressed heuristically by rejecting new hidden units that are too similar to existing ones. For example, the top two rows of the above table summarize the results of 10 learning trials on the two-spirals problem with and without such a heuristic.[3] By imposing a floor on the difference between two hidden units,[4] we reduced the size of our nets and the rate of generalization errors by 40%.

The following two-stage heuristic training method resulted in even better networks. During the first stage of learning we attempted to create a minimally necessary set of hidden units by searching only between training examples that were not yet divided by an existing hidden unit. During the second stage of learning we tried to increase the separability of our hidden codes by repeatedly computing an approximate set of output weights and then searching for hidden units between misclassified examples and nearby counterexamples. This heuristic was motivated by the observation that examples tend to be misclassified when a nearby piece of the target function's decision boundary has not been discovered. Ten trials of this method resulted in networks containing an average of just 54 hidden units, and the nets committed an average of only 29 mistakes on the test set. An example of a network generated by this method is shown in figure 3b.

For comparison, we made 10 attempts to train a 60-hidden-unit backprop net on the 2-spirals problem starting from uniform random weights and using conjugate gradient code provided by Steve Nowlan. While these nets had more than enough hidden units to compute the required function, not one of them succeeded in learning the complete training set.[5]

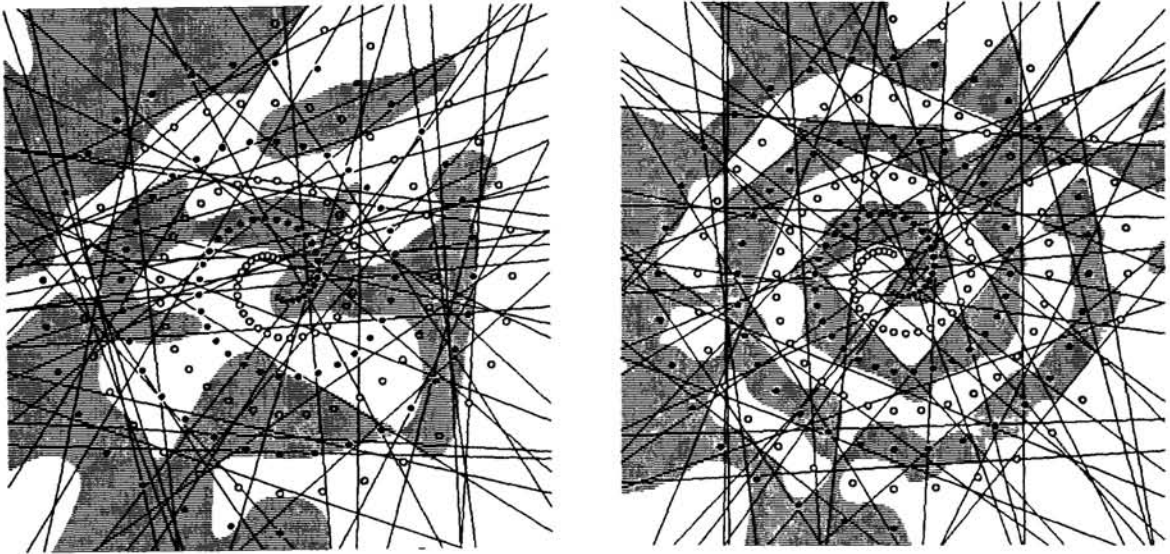

**Figure 4:** Backprop works better when started near a solution.

These results illustrate the main point of this paper: the currently prevalent training methodology (local optimization of random initial weights) is too weak to solve the NP-hard problem of hidden unit deployment. We believe that methods such as query learning which avoid the credit assignment problem are essential to the future of connectionism.

## Footnotes

[1] The additional points $p_i$ are obtained by perturbing $p_0$ with various small vectors $\pi_i$ and then diving back to the plane via a search that is slightly more complicated than the bisection method by which we found $p_0$. (Baum, 91) describes this search procedure in detail, as well as a technique for verifying that all the points $p_i$ lie on the *same* hidden plane.

[2]This "positive" and "negative" terminology suggests that the target net possesses a single output unit, but the method is not actually restricted to this case.

[3] To employ query learning, we defined the oracle function indicated by shading in figure 3a. The 194 training points are shown by dots in the figure. Our 576-element test set consisted of 3 points between each pair of adjacent same-class training points.

[4] Specifically, we required a minimum euclidean distance of 0.3 between the weights of two hidden units (after first normalizing the weight vectors so that the length of the non-threshold part of each vector was 1.

[5] Interestingly, a 2-50-1 backprop net whose initial weights were drawn from a handcrafted distribution (hidden units with uniform random positions together with the appropriate output weights) came much closer to success than 2-50-1 nets with uniform random initial weights (compare figures 4 and 2). We can sometimes address tough problems with backprop when our prior knowledge gives us a head start.

### References

E. Baum & D. Haussler. (1989) What size net gives valid generalization? *Neural Computation* **1**(1): 151-160.

E. Baum. (1991) *Neural Net Algorithms that Learn in Polynomial Time from Examples and Queries.* IEEE Transactions on Neural Networks **2**(1), January, 1991.

A. Blum & R. L. Rivest. (1988) Training a 3-node neural network is NP-complete. In D. S. Touretzky (ed.), *Advances in Neural Information Processing Systems 1,* 494-501. San Mateo, CA: Morgan Kaufmann.

K. Lang & M. Witbrock. (1988) *Learning to Tell Two Spirals Apart.* Proceedings of the 1988 Connectionist Models Summer School, Morgan Kaufmann.

D. Rumelhart, G. Hinton, & R. Williams. (1986) Learning internal representations by error propagation. In D. Rumelhart & J. McClelland (eds.) *Parallel Distributed Processing,* MIT Press.

L. G. Valiant. (1984) A theory of the learnable. *Comm. ACM* **27**(11): 1134-1142.
